# Policy Search via Density Estimation

**Andrew Y. Ng**
Computer Science Division
U.C. Berkeley
Berkeley, CA 94720
*ang@cs.berkeley.edu*

**Ronald Parr**
Computer Science Dept.
Stanford University
Stanford, CA 94305
*parr@cs.stanford.edu*

**Daphne Koller**
Computer Science Dept.
Stanford University
Stanford, CA 94305
*koller@cs.stanford.edu*

## Abstract

We propose a new approach to the problem of searching a space of stochastic controllers for a Markov decision process (MDP) or a partially observable Markov decision process (POMDP). Following several other authors, our approach is based on searching in parameterized families of policies (for example, via gradient descent) to optimize solution quality. However, rather than trying to estimate the values and derivatives of a policy directly, we do so indirectly using estimates for the probability densities that the policy induces on states at the different points in time. This enables our algorithms to exploit the many techniques for efficient and robust approximate density propagation in stochastic systems. We show how our techniques can be applied both to deterministic propagation schemes (where the MDP's dynamics are given explicitly in compact form,) and to stochastic propagation schemes (where we have access only to a generative model, or simulator, of the MDP). We present empirical results for both of these variants on complex problems.

## 1 Introduction

In recent years, there has been growing interest in algorithms for approximate planning in (exponentially or even infinitely) large Markov decision processes (MDPs) and partially observable MDPs (POMDPs). For such large domains, the value and $Q$-functions are sometimes complicated and difficult to approximate, even though there may be simple, compactly representable policies which perform very well. This observation has led to particular interest in *direct policy search* methods (e.g., [9, 8, 1]), which attempt to choose a good policy from some restricted class $\Pi$ of policies. In our setting, $\Pi = \{\pi_\theta : \theta \in \mathbb{R}^m\}$ is a class of policies smoothly parameterized by $\theta \in \mathbb{R}^m$. If the value of $\pi_\theta$ is differentiable in $\theta$, then gradient ascent methods may be used to find a locally optimal $\pi_\theta$. However, estimating values of $\pi_\theta$ (and the associated gradient) is often far from trivial. One simple method for estimating $\pi_\theta$'s value involves executing one or more Monte Carlo trajectories using $\pi_\theta$, and then taking the average empirical return; cleverer algorithms executing single trajectories also allow gradient estimates [9, 1]. These methods have become a standard approach to policy search, and sometimes work fairly well.

In this paper, we propose a somewhat different approach to this value/gradient estimation problem. Rather than estimating these quantities directly, we estimate the probability density over the states of the system induced by $\pi_\theta$ at different points in time. These *time slice*

*densities* completely determine the value of the policy $\pi_\theta$. While density estimation is not an easy problem, we can utilize existing approaches to density propagation [3, 5], which allow users to specify prior knowledge about the densities, and which have also been shown, both theoretically and empirically, to provide robust estimates for time slice densities. We show how direct policy search can be implemented using this approach in two very different settings of the planning problem: In the first, we have access to an explicit model of the system dynamics, allowing us to provide an explicit algebraic operator that implements the approximate density propagation process. In the second, we have access only to a generative model of the dynamics (which allows us only to sample from, but does not provide an explicit representation of, next-state distributions). We show how both of our techniques can be combined with gradient ascent in order to perform policy search, a somewhat subtle argument in the case of the sampling-based approach. We also present empirical results for both variants in complex domains.

## 2   Problem description

A *Markov Decision Process (MDP)* is a tuple $(S, s_0, A, R, P)$ where:[1] $S$ is a (possibly infinite) set of states; $s_0 \in S$ is a start state; $A$ is a finite set of actions; $R$ is a *reward function* $R : S \mapsto [0, R_{max}]$; $P$ is a *transition model* $P : S \times A \mapsto \Delta_S$, such that $P(s' \mid s, a)$ gives the probability of landing in state $s'$ upon taking action $a$ in state $s$.

A stochastic policy is a map $\pi : S \mapsto \Delta_A$, where $\pi(a \mid s)$ is the probability of taking action $a$ in state $s$. There are many ways of defining a policy $\pi$'s "quality" or *value*. For a horizon $T$ and discount factor $\gamma$, the *finite horizon discounted value function* $V_{T,\gamma}[\pi]$ is defined by
$$V_{0,\gamma}[\pi](s) = R(s) \,;\, V_{t+1,\gamma}[\pi](s) = R(s) + \gamma \sum_a \pi(a \mid s) \sum_{s'} P(s' \mid s, a) V_{t,\gamma}[\pi](s').$$
For an infinite state space (here and below), the summation is replaced by an integral. We can now define several optimality criteria. The *finite horizon total reward with horizon* $T$ is $V_T[\pi] = V_{T,1}[\pi](s_0)$. The *infinite horizon discounted reward with discount* $\gamma < 1$ is $V_\gamma[\pi] = \lim_{T \to \infty} V_{T,\gamma}[\pi](s_0)$. The *infinite horizon average reward* is $V_{avg}[\pi] = \lim_{T \to \infty} \frac{1}{T} V_{T,1}[\pi](s_0)$, where we assume that the limit exists.

Fix an optimality criterion $V$. Our goal is to find a policy that has a high value. As discussed, we assume we have a restricted set $\Pi$ of policies, and wish to select a good $\pi \in \Pi$. We assume that $\Pi = \{\pi_\theta \mid \theta \in \mathbb{R}^m\}$ is a set of policies parameterized by $\theta \in \mathbb{R}^m$, and that $\pi_\theta(a \mid s)$ is continuously differentiable in $\theta$ for each $s, a$. As a very simple example, we may have a one-dimensional state, two-action MDP with "sigmoidal" $\pi_\theta$, such that the probability of choosing action $a_0$ at state $x$ is $\pi_\theta(a_0 \mid x) = 1/(1 + \exp(-\theta_1 - \theta_2 x))$.

Note that this framework also encompasses cases where our family $\Pi$ consists of policies that depend only on certain aspects of the state. In particular, in POMDPs, we can restrict attention to policies that depend only on the observables. This restriction results in a subclass of stochastic memory-free policies. By introducing artificial "memory bits" into the process state, we can also define stochastic limited-memory policies. [6]

Each $\theta$ has a value $V[\theta] = V[\pi_\theta]$, as specified above. To find the best policy in $\Pi$, we can search for the $\theta$ that maximizes $V[\theta]$. If we can compute or approximate $V[\theta]$, there are many algorithms that can be used to find a local maximum. Some, such as *Nelder-Mead simplex search* (not to be confused with the simplex algorithm for linear programs), require only the ability to evaluate the function being optimized at any point. If we can compute or estimate $V[\theta]$'s gradient with respect to $\theta$, we can also use a variety of (deterministic or stochastic) *gradient ascent* methods.

## 3  Densities and value functions

Most optimization algorithms require some method for computing $V[\theta]$ for any $\theta$ (and sometimes also its gradient). In many real-life MDPs, however, doing so exactly is completely infeasible, due to the large or even infinite number of states. Here, we will consider an approach to estimating these quantities, based on a density-based reformulation of the value function expression. A policy $\pi$ induces a probability distribution over the states at each time $t$. Letting $\phi^{(0)}$ be the initial distribution (giving probability 1 to $s_0$), we define the *time slice distributions* via the recurrence:

$$\phi^{(t+1)}(s') = \sum_s \phi^{(t)}(s) \sum_a \pi(a \mid s) P(s' \mid s, a) \tag{1}$$

It is easy to verify that the standard notions of value defined earlier can reformulated in terms of $\phi^{(t)}$; e.g., $V_{T,\gamma}[\pi](s_0) = \sum_{t=0}^{T} \gamma^t(\phi^{(t)} \cdot R)$, where $\cdot$ is the dot-product operation (equivalently, the expectation of $R$ with respect to $\phi^{(t)}$). Somewhat more subtly, for the case of infinite horizon average reward, we have that $V_{avg}[\pi] = \phi^{(\infty)} \cdot R$, where $\phi^{(\infty)}$ is the limiting distribution of (1), if one exists.

This reformulation gives us an alternative approach to evaluating the value of a policy $\pi_\theta$: we first compute the time slice densities $\phi^{(t)}$ (or $\phi^{(\infty)}$), and then use them to compute the value. Unfortunately, that modification, by itself, does not resolve the difficulty. Representing and computing probability densities over large or infinite spaces is often no easier than representing and computing value functions. However, several results [3, 5] indicate that representing and computing high-quality *approximate* densities may often be quite feasible. The general approach is an approximate density propagation algorithm, using time-slice distributions in some restricted family $\Xi$. For example, in continuous spaces, $\Xi$ might be the set of multivariate Gaussians.

The approximate propagation algorithm modifies equation (1) to maintain the time-slice densities in $\Xi$. More precisely, for a policy $\pi_\theta$, we can view (1) as defining an operator $\Phi[\theta]$ that takes one distribution in $\Delta_S$ and returns another. For our current policy $\pi_{\theta_0}$, we can rewrite (1) as: $\phi^{(t+1)} = \Phi[\theta_0](\phi^{(t)})$. In most cases, $\Xi$ will not be closed under $\Phi$; approximate density propagation algorithms use some alternative operator $\hat{\Phi}$, with the properties that, for $\phi \in \Xi$: (a) $\hat{\Phi}(\phi)$ is also in $\Xi$, and (b) $\hat{\Phi}(\phi)$ is (hopefully) close to $\Phi(\phi)$. We use $\hat{\Phi}[\theta]$ to denote the approximation to $\Phi[\theta]$, and $\hat{\phi}^{(t)}$ to denote $(\hat{\Phi}[\theta])^{(t)}(\phi^{(0)})$. If $\hat{\Phi}$ is selected carefully, it is often the case that $\hat{\phi}^{(t)}$ is close to $\phi^{(t)}$. Indeed, a standard contraction analysis for stochastic processes can be used to show:

**Proposition 1** *Assume that for all $t$, $\|\Phi(\hat{\phi}^{(t)}) - \hat{\Phi}(\hat{\phi}^{(t)})\|_1 \leq \epsilon$. Then there exists some constant $\lambda$ such that for all $t$, $\|\hat{\phi}^{(t)} - \phi^{(t)}\|_1 \leq \epsilon/\lambda$.*

In some cases, $\lambda$ might be arbitrarily small, in which case the proposition is meaningless. However, there are many systems where $\lambda$ is reasonable (and independent of $\epsilon$) [3]. Furthermore, empirical results also show that approximate density propagation can often track the exact time slice distributions quite accurately.

Approximate tracking can now be applied to our planning task. Given an optimality criterion $V$ expressed with $\phi^{(t)}$s, we define an approximation $\hat{V}$ to it by replacing each $\phi^{(t)}$ with $\hat{\phi}^{(t)}$, e.g., $\hat{V}_{T,\gamma}[\pi](s_0) = \sum_{t=0}^{T} \gamma^t \hat{\phi}^{(t)} \cdot R$. Accuracy guarantees on approximate tracking induce comparable guarantees on the value approximation; from this, guarantees on the performance of a policy $\pi_{\hat{\theta}}$ found by optimizing $\hat{V}$ are also possible:

**Proposition 2** *Assume that, for all $t$, we have that $\|\hat{\phi}^{(t)} - \phi^{(t)}\|_1 \leq \delta$. Then for each fixed $T, \gamma$: $|V_{T,\gamma}[\pi](s_0) - \hat{V}_{T,\gamma}[\pi](s_0)| = O(\delta)$.*

**Proposition 3** *Let* $\theta^* = \arg\max_\theta V[\theta]$ *and* $\hat{\theta} = \arg\max_\theta \hat{V}[\theta]$. *If* $\max_\theta |V[\theta] - \hat{V}[\theta]| \leq \epsilon$, *then* $V[\theta^*] - V[\hat{\theta}] \leq 2\epsilon$.

## 4 Differentiating approximate densities

In this section we discuss two very different techniques for maintaining an approximate density $\hat{\phi}^{(t)}$ using an approximate propagation operator $\hat{\Phi}$, and show when and how they can be combined with gradient ascent to perform policy search. In general, we will assume that $\Xi$ is a family of distributions parameterized by $\xi \in \mathbb{R}^\ell$. For example, if $\Xi$ is the set of $d$-dimensional multivariate Gaussians with diagonal covariance matrices, $\xi$ would be a $2d$-dimensional vector, specifying the mean vector and the covariance matrix's diagonal.

Now, consider the task of doing gradient ascent over the space of policies, using some optimality criterion $\hat{V}$, say $\hat{V}_{T,\gamma}[\theta]$. Differentiating it relative to $\theta$, we get $\nabla_\theta \hat{V}_{T,\gamma}[\theta] = \sum_{t=0}^{T} \gamma^t \frac{d\hat{\phi}^{(t)}}{d\theta} \cdot R$. To avoid introducing new notation, we also use $\hat{\phi}^{(t)}$ to denote the associated vector of parameters $\xi \in \mathbb{R}^\ell$. These parameters are a function of $\theta$. Hence, the internal gradient term is represented by an $\ell \times m$ Jacobian matrix, with entries representing the derivative of a parameter $\xi_i$ relative to a parameter $\theta_j$. This gradient can be computed using a simple recurrence, based on the chain rule for derivatives:

$$\frac{d\hat{\phi}^{(t+1)}}{d\theta}(\theta_0) = \frac{d}{d\theta}\hat{\Phi}[\theta_0](\hat{\phi}^{(t)}) = \frac{\partial\hat{\Phi}}{\partial\theta}(\theta_0, \hat{\phi}^{(t)}) + \frac{\partial\hat{\Phi}}{\partial\hat{\phi}}(\theta_0, \hat{\phi}^{(t)}) \cdot \frac{d\hat{\phi}^{(t)}}{d\theta}(\theta_0). \quad (2)$$

The first summand (an $\ell \times m$ Jacobian) is the derivative of the transition operator $\hat{\Phi}$ relative to the policy parameters $\theta$. The second is a product of two terms: the derivative of $\hat{\Phi}$ relative to the distribution parameters, and the result of the previous step in the recurrence.

### 4.1 Deterministic density propagation

Consider a transition operator $\Phi$ (for simplicity, we omit the dependence on $\theta$). The idea in this approach is to try to get $\hat{\Phi}(\hat{\phi})$ to be as close as possible to $\Phi(\hat{\phi})$, subject to the constraint that $\hat{\Phi}(\hat{\phi}) \in \Xi$. Specifically, we define a *projection operator* $\Gamma$ that takes a distribution $\psi$ not in $\Xi$, and returns a distribution in $\Xi$ which is closest (in some sense) to $\psi$. We then define $\hat{\Phi}(\hat{\phi}) = \Gamma(\Phi(\hat{\phi}))$. In order to ensure that gradient descent applies in this setting, we need only ensure that $\Gamma$ and $\Phi$ are differentiable functions. Clearly, there are many instantiations of this idea for which this assumption holds. We provide two examples.

Consider a continuous-state process with nonlinear dynamics, where $\Phi$ is a mixture of conditional linear Gaussians. We can define $\Xi$ to be the set of multivariate Gaussians. The operator $\Gamma$ takes a distribution (a mixture of gaussians) $\psi$ and computes its mean and covariance matrix. This can be easily computed from $\psi$'s parameters using simple differentiable algebraic operations.

A very different example is the algorithm of [3] for approximate density propagation in *dynamic Bayesian networks (DBNs)*. A DBN is a structured representation of a stochastic process, that exploits conditional independence properties of the distribution to allow compact representation. In a DBN, the state space is defined as a set of possible assignments $x$ to a set of random variables $X_1, \ldots, X_n$. The transition model $P(x' \mid x)$ is described using a Bayesian network fragment over the nodes $\{X_1, \ldots, X_n, X'_1, \ldots, X'_n\}$. A node $X_i$ represents $X_i^{(t)}$ and $X'_i$ represents $X_i^{(t+1)}$. The nodes $X_i$ in the network are forced to be *roots* (i.e., have no parents), and are not associated with conditional probability distributions. Each node $X'_i$ is associated with a conditional probability distribution (CPD), which specifies $P(X'_i \mid \text{Parents}(X'_i))$. The transition probability $P(\boldsymbol{X'} \mid \boldsymbol{X})$ is defined as

$\prod_i P(X_i' \mid \text{Parents}(X_i'))$. DBNs support a compact representation of complex transition models in MDPs [2]. We can extend the DBN to encode the behavior of an MDP with a stochastic policy $\pi$ by introducing a new random variable $A$ representing the action taken at the current time. The parents of $A$ will be those variables in the state on which the action is allowed to depend. The CPD of $A$ (which may be compactly represented with function approximation) is the distribution over actions defined by $\pi$ for the different contexts.

In discrete DBNs, the number of states grows exponentially with the number of state variables, making an explicit representation of a joint distribution impractical. The algorithm of [3] defines $\Xi$ to be a set of distributions defined compactly as a set of marginals over smaller clusters of variables. In the simplest example, $\Xi$ is the set of distributions where $X_1, \ldots, X_n$ are independent. The parameters $\xi$ defining a distribution in $\Xi$ are the parameters of $n$ multinomials. The projection operator $\Gamma$ simply marginalizes distributions onto the individual variables, and is differentiable. One useful corollary of [3]'s analysis is that the decay rate of a structured $\hat{\Phi}$ over $\Xi$ can often be much higher than the decay rate of $\Phi$, so that multiple applications of $\hat{\Phi}$ can converge very rapidly to a stationary distribution; this property is very useful when approximating $\phi^{(\infty)}$ to optimize relative to $V_{avg}$.

## 4.2 Stochastic density propagation

In many settings, the assumption that we have direct access to $\Phi$ is too strong. A weaker assumption is that we have access to a *generative model* — a black box from which we can generate samples with the appropriate distribution; i.e., for any $s, a$, we can generate samples $s'$ from $P(s' \mid s, a)$. In this case, we use a different approximation scheme, based on [5]. The operator $\hat{\Phi}$ is a stochastic operator. It takes the distribution $\hat{\phi}$, and generates some number of random state samples $s_i$ from it. Then, for each $s_i$ and each action $a$, we generate a sample $s_i'$ from the transition distribution $P(\cdot \mid s_i, a)$. This sample $\langle s_i, a_i, s_i' \rangle$ is then assigned a weight $w_i = \pi_\theta(a_i \mid s_i)$, to compensate for the fact that not all actions would have been selected by $\pi_\theta$ with equal probability. The resulting set of $N$ samples $s_i'$ weighted by the $w_i$s is given as input to a statistical density estimator, which uses it to estimate a new density $\hat{\phi}'$. We assume that the density estimation procedure is a differentiable function of the weights, often a reasonable assumption.

Clearly, this $\hat{\Phi}$ can be used to compute $\hat{\phi}^{(t)}$ for any $t$, and thereby approximate $\pi_\theta$'s value. However, the gradient computation for $\hat{\Phi}$ is far from trivial. In particular, to compute the derivative $\partial \hat{\Phi} / \partial \hat{\phi}$, we must consider $\hat{\Phi}$'s behavior for some perturbed $\hat{\phi}_1^{(t)}$ other than the one (say, $\hat{\phi}_0^{(t)}$) to which it was applied originally. In this case, an entirely different set of samples would probably have been generated, possibly leading to a very different density. It is hard to see how one could differentiate the result of this perturbation. We propose an alternative solution based on *importance sampling*. Rather than change the samples, we modify their weights to reflect the change in the probability that they would be generated. Specifically, when fitting $\hat{\phi}_1^{(t+1)}$, we now define a sample $\langle s_i, a_i, s_i' \rangle$'s weight to be

$$w_i(\hat{\phi}_1^{(t)}, \boldsymbol{\theta}) = \frac{\hat{\phi}_1^{(t)}(s_i) \pi_\theta(a_i \mid s_i)}{\hat{\phi}_0^{(t)}(s_i)}. \tag{3}$$

We can now compute $\hat{\Phi}$'s derivatives at $(\boldsymbol{\theta}_0, \hat{\phi}_0^{(t)})$ with respect to any of its parameters, as required in (2). Let $\zeta$ be the vector of parameters $(\boldsymbol{\theta}, \boldsymbol{\xi})$. Using the chain rule, we have

$$\frac{\partial \hat{\Phi}[\boldsymbol{\theta}](\hat{\phi})}{\partial \zeta} = \frac{\partial \hat{\Phi}[\boldsymbol{\theta}](\hat{\phi})}{\partial \boldsymbol{w}} \cdot \frac{\partial \boldsymbol{w}}{\partial \zeta}.$$

The first term is the derivative of the estimated density relative to the sample weights (an $\ell \times N$ matrix). The second is the derivative of the weights relative to the parameter vector (an $N \times (m + \ell)$ Jacobian), which can easily be computed from (3).

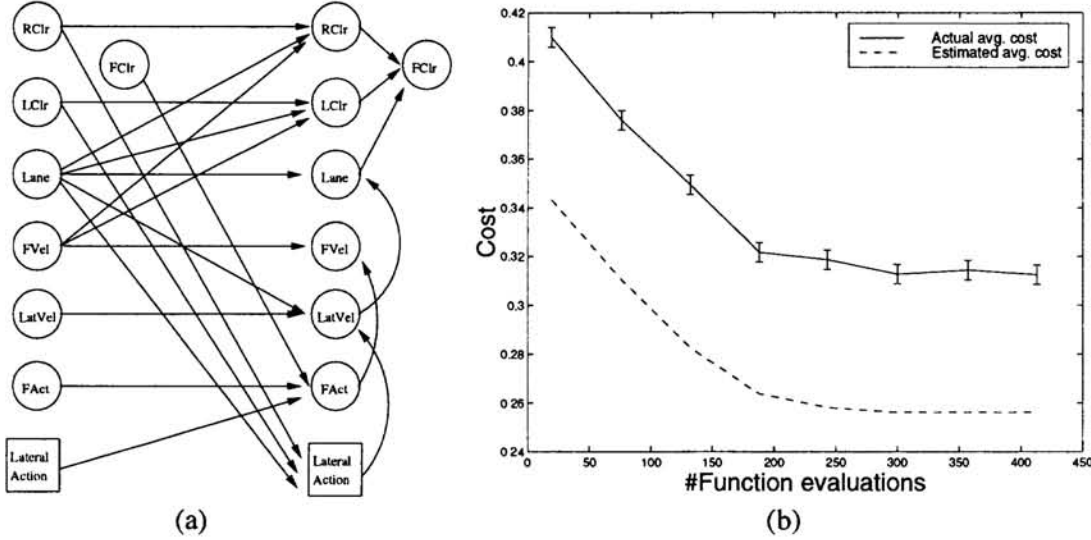

Figure 1: Driving task: (a) DBN model; (b) policy-search/optimization results (with 1 s.e.)

## 5   Experimental results

We tested our approach in two very different domains. The first is an average-reward DBN-MDP problem (shown in Figure 1(a)), where the task is to find a policy for changing lanes when driving on a moderately busy two-lane highway with a slow lane and a fast lane. The model is based on the BAT DBN of [4], the result of a separate effort to build a good model of driver behavior. For simplicity, we assume that the car's speed is controlled automatically, so we are concerned only with choosing the *Lateral Action – change lane* or *drive straight*. The observables are shown in the figure: *LClr* and *RClr* are the clearance to the next car in each lane (*close, medium* or *far*). The agent pays a cost of 1 for each step it is "blocked" by (meaning driving *close* to) the car to its front; it pays a penalty of 0.2 per step for staying in the fast lane. Policies are specified by action probabilities for the 18 possible observation combinations. Since this is a reasonably small number of parameters, we used the simplex search algorithm described earlier to optimize $\hat{V}[\theta]$.

The process mixed quite quickly, so $\hat{\phi}^{(20)}$ was a fairly good approximation to $\hat{\phi}^{(\infty)}$. $\Xi$ used a fully factored representation of the joint distribution except for a single cluster over the three observables. Evaluations are averages of 300 Monte Carlo trials of 400 steps each. Figure 1(b) shows the estimated and actual average rewards, as the policy parameters are evolved over time. The algorithm improved quickly, converging to a very natural policy with the car generally staying in the slow lane, and switching to the fast lane only when necessary to overtake.

In our second experiment, we used the bicycle simulator of [7]. There are 9 actions corresponding to leaning left/center/right and applying negative/zero/positive torque to the handlebar; the six-dimensional state used in [7] includes variables for the bicycle's tilt angle and orientation, and the handlebar's angle. If the bicycle tilt exceeds $\pi/15$, it falls over and enters an absorbing state. We used policy search over the following space: we selected twelve (simple, manually chosen but not fine-tuned) features of each state; actions were chosen with a softmax — the probability of taking action $a_i$ is $\exp(x \cdot w_i)/\sum_j \exp(x \cdot w_j)$. As the problem only comes with a generative model of the complicated, nonlinear, noisy bicycle dynamics, we used the stochastic density propagation version of our algorithm, with (stochastic) gradient ascent. Each distribution in $\Xi$ was a mixture of a singleton point consisting of the absorbing-state, and of a 6-D multivariate Gaussian.

The first task in this domain was to balance reliably on the bicycle. Using a horizon of $T = 200$, discount $\gamma = 0.995$, and 600 $s_i$ samples per density propagation step, this was quickly achieved. Next, trying to learn to ride to a goal[2] 10m in radius and 1000m away, it also succeeded in finding policies that do so reliably. Formal evaluation is difficult, but this is a sufficiently hard problem that even finding a solution can be considered a success. There was also some slight parameter sensitivity (and the best results were obtained only with $\hat{\phi}^{(0)}$ picked/fit with some care, using in part data from earlier and less successful trials, to be "representative" of a fairly good rider's state distribution,) but using this algorithm, we were able to obtain solutions with median riding distances under 1.1km to the goal. This is significantly better than the results of [7] (obtained in the learning rather than planning setting, and using a value-function approximation solution), which reported much larger riding distances to the goal of about 7km, and a single "best-ever" trial of about 1.7km.

## 6  Conclusions

We have presented two new variants of algorithms for performing direct policy search in the deterministic and stochastic density propagation settings. Our empirical results have also shown these methods working well on two large problems.

**Acknowledgements.** We warmly thank Kevin Murphy for use of and help with his Bayes Net Toolbox, and Jette Randløv and Preben Alstrøm for use of their bicycle simulator. A. Ng is supported by a Berkeley Fellowship. The work of D. Koller and R. Parr is supported by the ARO-MURI program "Integrated Approach to Intelligent Systems", DARPA contract DACA76-93-C-0025 under subcontract to IET, Inc., ONR contract N66001-97-C-8554 under DARPA's HPKB program, the Sloan Foundation, and the Powell Foundation.

## Footnotes

[1] We write rewards as $R(s)$ rather than $R(s, a)$, and assume a single start state rather than an initial-state distribution, only to simplify exposition; these and several other minor extensions are trivial.

[2]For these experiments, we found learning could be accomplished faster with the simulator's integration delta-time constant tripled for training. This and "shaping" reinforcements (chosen to reward progress made towards the goal) were both used, and training was with the bike "infinitely distant" from the goal. For this and the balancing experiments, sampling from the fallen/absorbing-state portion of the distributions $\hat{\phi}^{(t)}$ is obviously inefficient use of samples, so all samples were drawn from the non-absorbing state portion (i.e. the Gaussian, also with its tails corresponding to tilt angles greater than $\pi/15$ truncated), and weighted accordingly relative to the absorbing-state portion.

## References

[1] L. Baird and A.W. Moore. Gradient descent for general Reinforcement Learning. In *NIPS 11*, 1999.

[2] C. Boutilier, T. Dean, and S. Hanks. Decision theoretic planning: Structural assumptions and computational leverage. *J. Artificial Intelligence Research*, 1999.

[3] X. Boyen and D. Koller. Tractable inference for complex stochastic processes. In *Proc. UAI*, pages 33–42, 1998.

[4] J. Forbes, T. Huang, K. Kanazawa, and S.J. Russell. The BATmobile: Towards a Bayesian automated taxi. In *Proc. IJCAI*, 1995.

[5] D. Koller and R. Fratkina. Using learning for approximation in stochastic processes. In *Proc. ICML*, pages 287–295, 1998.

[6] N. Meuleau, L. Peshkin, K-E. Kim, and L.P. Kaelbling. Learning finite-state controllers for partially observable environments. In *Proc. UAI 15*, 1999.

[7] J. Randløv and P. Alstrøm. Learning to drive a bicycle using reinforcement learning and shaping. In *Proc. ICML*, 1998.

[8] J.K. Williams and S. Singh. Experiments with an algorithm which learns stochastic memoryless policies for POMDPs. In *NIPS 11*, 1999.

[9] R.J. Williams. Simple statistical gradient-following algorithms for connectionist reinforcement learning. *Machine Learning*, 8:229–256, 1992.

